# Structural inference affects depth perception in the context of potential occlusion

**Ian H. Stevenson and Konrad P. Körding**
Department of Physical Medicine and Rehabilitation
Northwestern University
Chicago, IL 60611
`i-stevenson@northwestern.edu`

## Abstract

In many domains, humans appear to combine perceptual cues in a near-optimal, probabilistic fashion: two noisy pieces of information tend to be combined linearly with weights proportional to the precision of each cue. Here we present a case where structural information plays an important role. The presence of a background cue gives rise to the possibility of occlusion, and places a soft constraint on the location of a target - in effect propelling it forward. We present an ideal observer model of depth estimation for this situation where structural or ordinal information is important and then fit the model to human data from a stereo-matching task. To test whether subjects are truly using ordinal cues in a probabilistic manner we then vary the uncertainty of the task. We find that the model accurately predicts shifts in subject's behavior. Our results indicate that the nervous system estimates depth ordering in a probabilistic fashion and estimates the structure of the visual scene during depth perception.

## 1 Introduction

Understanding how the nervous system makes sense of uncertain visual stimuli is one of the central goals of perception research. One strategy to reduce uncertainty is to combine cues from several sources into a good joint estimate. If the cues are Gaussian, for instance, an ideal observer should combine them linearly with weights proportional to the precision of each cue. In the past few decades, a number of studies have demonstrated that humans combine cues during visual perception to reduce uncertainty and often do so in near-optimal, probabilistic ways [1, 2, 3, 4].

In most situations, each cue gives noisy information about the variable of interest that can be modeled as a Gaussian likelihood function about the variable. Recently [5] have suggested that subjects may combine a metric cue (binocular disparity) with ordinal cues (convexity or familiarity of faces) during depth perception. In these studies ordinal cues were modeled as simple biases. We argue that the effect of such ordinal cues stems from a structural inference process where an observer estimates the structure of the visual scene along with depth cues.

The importance of structural inference and occlusion constraints, particularly of hard constraints, has been noted previously [6, 7, 8]. For instance, it was found that points presented to one eye but not the other have a perceived depth that is constrained by the position of objects presented to both eyes. Although these unpaired image points do not contain depth cues in the usual sense, subjects were able to estimate their depth. This indicates that human subjects indeed use the inferred structure of a visual scene for the estimation of depth.

Here we formalize the constraints presented by occlusion using a probabilistic framework. We first present the model and illustrate its ability to describe data from [7]. Then we present results from a new stereo-vision experiment in which subjects were asked to match the depth of an occluding

or occluded circle. The model accurately predicts human behavior in this task and describes the changes that occur when we increase depth uncertainty. These results cannot be explained by traditional cue combination or even more recent relevance (causal inference) models [9, 10, 11, 12]. Our constraint-based approach may thus be useful in understanding how subjects make sense of cluttered scenes and the impact of structural inference on perception.

## 2 Theory

### 2.1 An Ordinal Cue Combination Model

We assume that observers receive noisy information about the depth of objects in the world. For concreteness, we assume that there is a central object $c$ and a surrounding object $s$. The exact shapes and relative positions of these two objects are not important, but naming them will simplify the notation that follows. We assume that each of these objects has a true, hidden depth ($x_c$ and $x_s$) and observers receive noisy observations of these depths ($y_c$ and $y_s$).

In a scene with potential occlusion there may be two (or more) possible interpretations of an image (Fig 1A). When there is no occlusion (structure $S_1$) the depth observations of the two objects are independent. That is, we assume that the depth of the surrounding object in the scene $s$ has no influence on our estimate of the depth of $c$. The distribution of observations is assumed to be Gaussian and is physically determined by disparity, shading, texture, or other depth cues and their associated uncertainties. In this case the joint distribution of the observations given the hidden positions is

$$p(y_c, y_s | x_c, x_s, S_1) = p(y_c | x_c, S_1)p(y_s | x_s, S_1) = N_{y_c}(x_c, \sigma_c)N_{y_s}(x_s, \sigma_s). \tag{1}$$

When occlusion does occur, however, the position of the central object $c$ is bounded by the depth of the surrounding, occluded object (structure $S_2$)

$$p(y_c, y_s | x_c, x_s, S_2) \propto \begin{cases} N_{y_c}(x_c, \sigma_c)N_{y_s}(x_s, \sigma_s) & \text{if } x_c > x_s, \\ 0 & \text{if } x_c \leq x_s. \end{cases} \tag{2}$$

An ideal observer can then make use of this ordinal information in estimating the depth of the occluding object. The (marginal) posterior distribution over the hidden depth of the central object $x_c$ can be found by marginalizing over the depth of the surrounding object $x_s$ and possible structures ($S_1$ and $S_2$).

$$p(x_c \mid y_c, y_s) = p(x_c \mid y_c, y_s, S_1)p(S_1) + p(x_c \mid y_c, y_s, S_2)p(S_2) \tag{3}$$

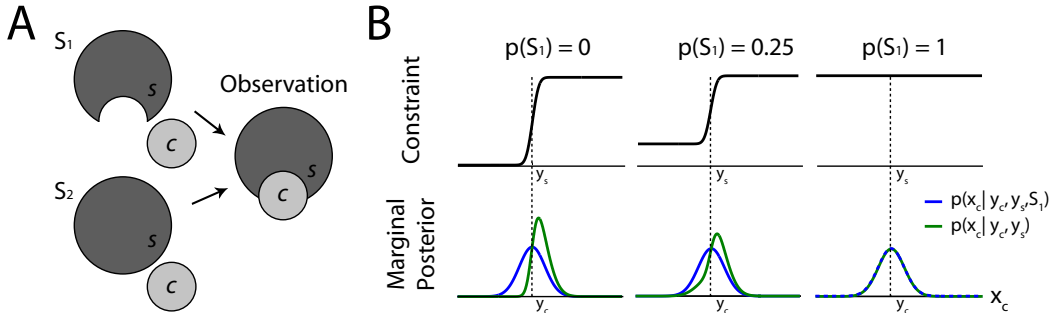

Figure 1: An occlusion model with soft-constraints. (A) Two possible structures leading to the same observation: one without occlusion $S_1$ and one with occlusion $S_2$. (B) Examples of biases in the posterior estimate of $x_c$ for complete (left), moderate (center), and no relevance (right). In the cases shown, the observed depth of the central stimulus $y_c$ is the same as the observed depth of the surrounding stimulus $y_s$. Note that when $y_c \gg y_s$ the constraint will not bias estimates of $x_c$.

Using the assumption of conditional independence and assuming flat priors over the hidden depths $x_c$ and $x_s$, the first term in this expression is

$$p(x_c \mid y_c, y_s, S_1) = \int p(x_c|y_c, y_s, x_s, S_1)p(x_s \mid y_c, y_s, S_1)dx_s$$
$$= \int p(x_c|y_c, S_1)p(x_s|y_s, S_1)dx_s = \int N_{x_c}(y_c, \sigma_c)N_{x_s}(y_s, \sigma_s)dx_s \quad (4)$$
$$= N_{x_c}(y_c, \sigma_c).$$

The second term is then

$$p(x_c \mid y_c, y_s, S_2) = \int p(x_c|y_c, y_s, x_s, S_2)p(x_s \mid y_c, y_s, S_2)dx_s$$
$$= \int p(y_c, y_s|x_c, x_s, S_2)dx_s$$
$$= \int_{-\infty}^{x_c} N_{x_c}(y_c, \sigma_c)N_{x_s}(y_s, \sigma_s)dx_s \quad (5)$$
$$= \frac{1}{Z}[erf(\rho_s(x_c - y_s))/2 + 1/2]N_{x_c}(y_c, \sigma_c),$$

where step 2 uses Bayes' rule and the assumption of flat priors, $\rho_s = 1/\sqrt{(2\pi)}/\sigma_s$ and $Z$ is a normalizing factor. Combining these two terms gives the marginal posterior

$$p(x_c \mid y_c, y_s) = \frac{1}{Z}\left[(1 - p(S_1))(erf(\rho_s(x_c - y_s))/2 + 1/2) + p(S_1)\right] N_{x_c}(y_c, \sigma_c), \quad (6)$$

which describes the best estimate of the depth of the central object. Intuitively, the term in square brackets constrains the possible depths of the central object $c$ (Fig 1B). The $p(S_1)$ term allows for the possibility that the constraint should not apply. Similar to models of causal inference [11, 12, 9, 10], the surrounding stimulus may be irrelevant, in which case we should simply rely on the observation of the target.

Here we have described two specific structures in the world that result in the same observation. Real world stimuli may result from a much larger set of possible structures. Generally, we can simply split structures into those with occlusion $O$ and those without occlusion $\neg O$. Above, $S_1$ corresponds to the set of possible structures without occlusion $\neg O$, and $S_2$ corresponds to the set of possible structures with occlusion $O$. It is not necessary to actually enumerate the possible structures.

Similar to traditional cue combination models, where there is an analytic form for the expected value of the target (linear combination weighted by the precision of each cue), we can write down analytic expressions for $E[x_c]$ for at least one case. For $p(S_1) = 0$, $\sigma_s \to 0$ the mean of the marginal posterior is the expected value of a truncated Gaussian

$$E(x_c|y_s < x_c) = y_c + \sigma_c\lambda(\frac{y_s - y_c}{\sigma_c}) \quad (7)$$

Where $\lambda(\cdot) = \frac{\phi(\cdot)}{[1-\Phi(\cdot)]}$, $\phi(\cdot)$ is the PDF for the standard normal distribution and $\Phi(\cdot)$ is the CDF. For $y_c = y_s$, for instance,

$$E(x_c|y_s < x_c) = y_c + 0.8\sigma_c \quad (8)$$

It is important to note that, similar to classical cue combination models, estimation of the target is improved by combining depth information with the occlusion constraint. The variance of $p(x_c|y_c, y_s)$ is smaller than that of $p(x_c \mid y_c, y_s, S_1)$.

## 2.2  Modeling Data from Nakayama and Shimojo (1990)

To illustrate the utility of this model, we fit data from [7]. In this experiment subjects were presented with a rectangle in each eye. Horizontal disparity between the two rectangles gave the impression of depth. To test how subjects perceive occluded objects, a small vertical bar was presented to one eye, giving the impression that the large rectangle was occluding the bar and leading to unpaired image points (Fig 2A). Subjects were then asked to match the depth of this vertical bar by changing the disparity of another image in which the bar was presented in stereo. Despite the absence of direct depth cues, subjects assigned a depth to the vertical bar. Moreover, for a range of horizontal distances, the assigned depth was consistent with the constraint provided by the stereo-rectangle (Fig 2B). These results systematically characterize the effect of structural estimation on depth estimates. Without ordinal information, the horizontal distance between the rectangle and the vertical bar should have no effect on the perceived depth of the bar.

In our model $y_c$ and $y_s$ are simply observations on the depth of two objects: in this case, the unpaired vertical bar and the large rectangle. Since there isn't direct disparity for the vertical bar, we assume that horizontal distance from the large rectangle serves as the depth cue. In reality an infinity of depths are compatible with a given horizontal distance (Fig 2A, dotted lines). However, the size and shape of the vertical bar serve as indirect cues, which we assume generate a Gaussian likelihood (as in Eq. 1). We fit our model to this data with three free parameters: $\sigma_s$, $\sigma_c$, and a relevance term $p(O)$. The event $O$ corresponds to occlusion (case $S_2$), while $\neg O$ corresponds to the set of possible structures leading to the same observation without occlusion. For the valid stimuli where occlusion can account for the vertical bar being seen in only one eye, $\sigma_s = 4.45$ arcmin, $\sigma_c = 12.94$ arcmin and $p(\neg O) = 0.013$ minimized the squared error between the data and model fit (Fig 2C). For invalid stimuli we assume that $p(\neg O) = 1$, which matches subject's responses.

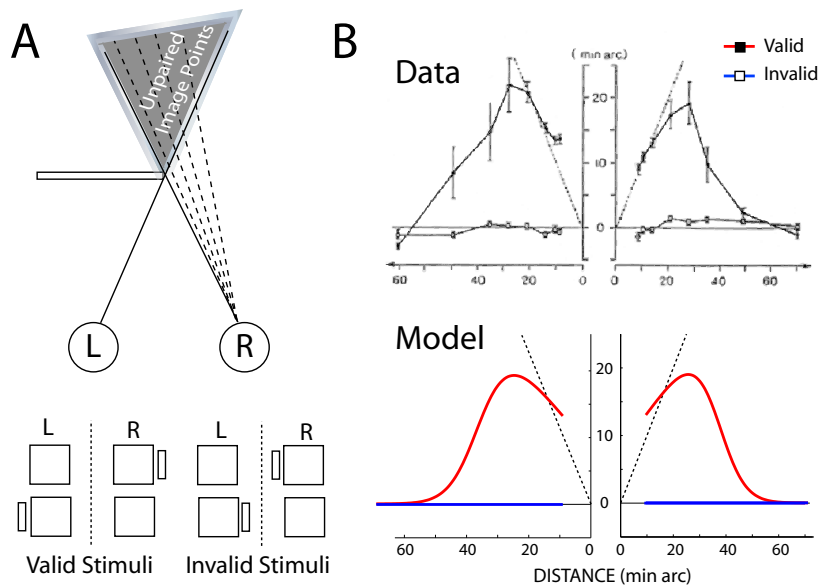

Figure 2: Experiment and data from [7]. A) Occlusion puts hard constraints on the possible depth of unpaired image points (top). This leads to "valid" and "invalid" stimuli (bottom). B) When subjects were asked to judge the depth of unpaired image points they followed these hard constraints (dotted lines) for a range of distances between the large rectangle and vertical bar (top). The two figures show a single subject's response when the vertical bar was positioned to the left or right of a large rectangle. The ordinal cue combination model can describe this behavior as well as deviations from the constraints for large distances (bottom).

## 3 Experimental Methods

To test this model in a more general setting where depth is driven by both paired and unpaired image points we constructed a simple depth matching experiment. Subjects (N=7) were seated 60cm in front of a CRT wearing shutter glasses (StereoGraphics CrystalEyes, 100Hz refresh rate) and asked to maintain their head position on a chin-rest. The experiment consisted of two tasks: a two-alternative forced choice task (2AFC) to measure subjects' depth acuity and a stereo-matching task to measure their perception of depth when a surrounding object was present. The target (central) objects were drawn on-screen as circles (13.0 degrees diameter) composed of random dots on a background pedestal of random dots (Fig 3).

In the 2AFC task, subjects were presented with two target objects with slightly different horizontal disparities and asked to indicate using the keyboard which object was closer. The reference object had a horizontal disparity of 0.57 degrees and was positioned randomly each trial on either the left or right side. The pedestal had a horizontal disparity of -0.28 degrees. Subjects performed 100 trials in which the disparity of the test object was chosen using optimal experimental design methods [13]. After the first 10 trials the next sample was chosen to maximize the conditional mutual information between the responses and the parameter for the just-noticeable depth difference (JND) given the sample position. This allowed us to efficiently estimate the JND for each subject.

In the stereo-matching task subjects were presented with two target objects and a larger surrounding circle (25.2 degrees diameter) paired with one of the targets. Subjects were asked to match the depth of the unpaired target with that of the paired target using the keyboard (100 trials). The depth of the paired target was held fixed across trials at 0.57 degrees horizontal disparity while the position of the surrounding circle was varied between 0.14-1.00 degrees horizontal disparity. The depth of the unpaired target was selected randomly at the beginning of each trial to minimize any effects of the starting position. All objects were presented in gray-scale and the target was presented off-center from the surrounding object to avoid confounding shape cues. The side on which the paired target and surrounding object appeared (left or right side of the screen) was also randomly chosen from trial to trial, and all objects were within the fusional limits for this task. When asked, subjects reported that diplopia occurred only when they drove the unpaired target too far in one direction or the other.

Each of these tasks (the 2AFC task and the stereo-matching task) was performed for two uncertainty conditions: a low and high uncertainty condition. We varied the uncertainty by changing the distribution of disparities for the individual dots which composed the target objects and the larger occluding/occluded circle. In the low uncertainty condition the disparity for each dot was drawn from a Gaussian distribution with a variance of 2.2 arc minutes. In the high uncertainty condition

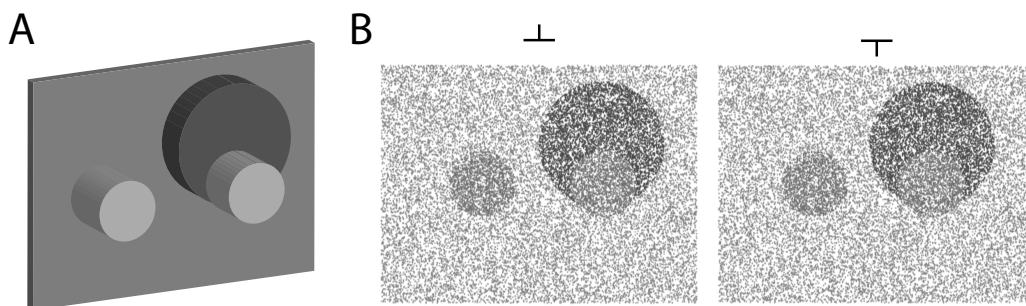

Figure 3: Experimental design. Each trial consists of a matching task in which subjects control the depth of an unpaired circle (A, left). Subjects attempt to match the depth of this unpaired circle to the depth of a target circle which is surrounded by a larger object (A, right). Divergent fusers can fuse (B) to see the full stimulus. The contrast has been reversed for visibility. To measure depth acuity, subjects also complete a two-alternative forced choice task (2AFC) using the same stimulus without the surrounding object.

the disparities were drawn with a variance of 6.5 arc minutes. All subjects had normal or corrected to normal vision and normal stereo vision (as assessed by a depth acuity $< 5$ arcmin in the low uncertainty 2AFC task). All experimental protocols were approved by IRB and in accordance with Northwestern University's policy statement on the use of humans in experiments. Informed consent was obtained from all participants.

## 4 Results

All subjects showed increased just-noticeable depth differences between the low and high uncertainty conditions. The JNDs were significantly different across conditions (one-sided paired t-test, p= 0.0072), suggesting that our manipulation of uncertainty was effective (Fig 4A). In the matching task, subjects were, on average, biased by the presence of the surrounding object. As the disparity of the surrounding object was increased and disparity cues suggested that $s$ was closer than $c$, this bias increased. Consistent with our model, this bias was higher in the high uncertainty condition (Fig 4B and C). However, the difference between uncertainty conditions was only significant for two surround depths (0.6 and 1.0 degrees, one-sided paired t-test p=0.004, p=0.0281) and not significant as a main effect (2-way ANOVA p=0.3419). To model the bias, we used the JNDs estimated from the 2AFC task, and fit two free parameters: $\sigma_s$ and $p(\neg O)$, by minimizing the squared error between model predictions and subject's responses. The model provided an accurate fit for both individual subjects and the across subject data (Fig 4B and C). For the across subject data, we found $\sigma_s = 0.085$ arcmin for the low uncertainty condition and $\sigma_s = 0.050$ arcmin for the high uncertainty

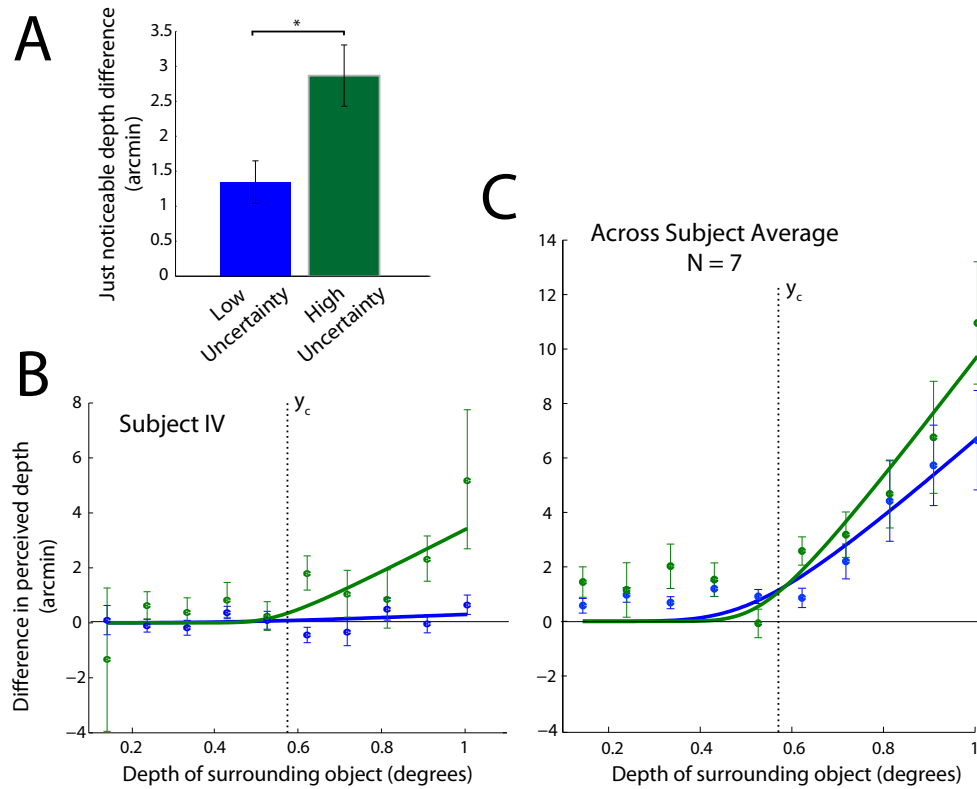

Figure 4: Experimental results. (A) Just noticeable depth differences for the two uncertainty conditions averaged across subjects. (B) and (C) show the difference between the perceived depth of the unpaired target and the paired target (the bias) as a function of the depth of the surrounding circle. Results for a typical subject (B) and the across subject average (C). Dots and error-bars denote subject responses, solid lines denote model fits, and dotted lines denote the depth of the paired target, which was fixed. Error bars denote SEM (N=7).

condition. In these cases, $p(\neg O)$ was not significantly different from zero and the simplified model in which $p(\neg O) = 0$ was preferred (cross-validated likelihood ratio test). Over the range of depths we tested, this relevance term does not seem to play a role. However, we predict that for larger discrepancies this relevance term would come into play as subjects begin to ignore the surrounding object (as in Fig 2).

Note that if the presence of a surrounding object had no effect subjects would be unbiased across depths of the occluded object. Two subjects (out of 7) did not show bias; however, both subjects had normal stereo vision and this behavior did not appear to be correlated with low or high depth acuity. Since subjects were allowed to free-view the stimulus, it is possible that some subjects were able to ignore the surrounding object completely. As with the invalid stimuli in [7], a model where $p(\neg O) = 1$ accurately fit data from these subjects. The rest of the subjects demonstrated bias (see Fig 4B for an example), but more data may be need to conclusively show differences between the two uncertainty conditions and causal inference effects.

## 5 Discussion

The results presented above illustrate the importance of structural inference in depth perception. We have shown that potential occlusion can bias perceived depth, and a probabilistic model of the constraints accurately accounts for subjects' perception during occlusion tasks with unpaired image points [7] as well as a novel task designed to probe the effects of structural inference.

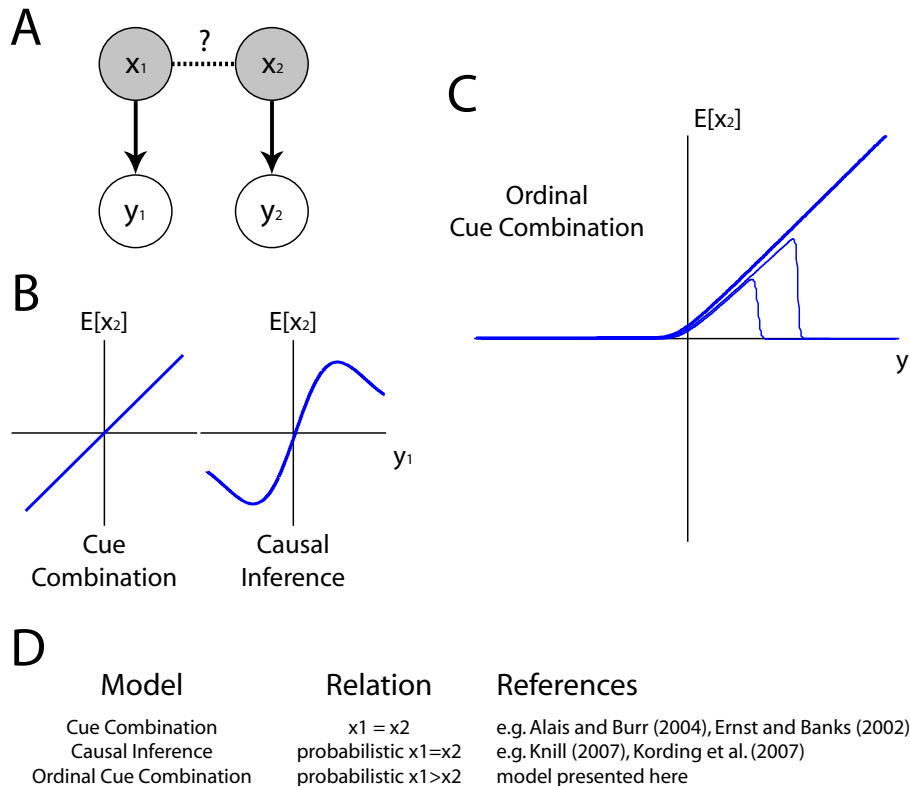

Figure 5: Models of cue combination. (A) Given the observations ($y_1$ and $y_2$) from two sources, how should we estimate the hidden sources $x_1$ and $x_2$? (B) Classical cue combination models assume $x_1 = x_2$. This results in a linear weighting of the cues. Non-linear cue combination can be explained by causal inference models where $x_1$ and $x_2$ are probabilistically equal. (C) In the model presented here, ordinal information introduces an asymmetry into cue combination. $x_1$ and $x_2$ are related here by a probabilistic inequality. (D) A summary of the relation between $x_1$ and $x_2$ for each model class.

A number of studies have proposed probabilistic accounts of depth perception [1, 4, 12, 14], and a variety of cues, such as disparity, shading, and texture, can all be combined to estimate depth [4, 12]. However, accounting for structure in the visual scene and use of occlusion constraints is typically qualitative or limited to hard constraints where certain depth arrangements are strictly ruled out [6, 14]. The model presented here accounts for a range of depth perception effects including perception of both paired and unpaired image points. Importantly, this model of perception explains the effects of ordinal cues in a cohesive structural inference framework.

More generally, ordinal information introduces asymmetry into cue combination. Classically, cue combination models assume a generative model in which two observations arise from the same hidden source. That is, the hidden source for observation 1 is equal to the hidden source for observation 2 (Fig 5A). More recently, causal inference or cue conflict models have been developed that allow for the possibility of probabilistic equality [9, 11, 12]. That is, there is some probability that the two sources are equal and some probability that they are unequal. This addition explains a number of nonlinear perceptual effects [9, 10] (Fig 5B). The model presented here extends these previous models by introducing ordinal information and allowing the relationship between the two sources to be an inequality - where the value from one source is greater than or less than the other. As with causal inference models, relevance terms allow the model to capture probabilistic inequality, and this type of mixture model allows descriptions of asymmetric and nonlinear behavior (Fig 5C). The ordinal cue combination model thus increases the class of behaviors that can be modeled by cue combination and causal inference and should have applications for other modalities where ordinal and structural information is important.

# References

[1] M. O. Ernst and M. S. Banks. Humans integrate visual and haptic information in a statistically optimal fashion. *Nature*, 415(6870):429–33, 2002.

[2] D. Kersten and A. Yuille. Bayesian models of object perception. *Current Opinion in Neurobiology*, 13(2):150–158, 2003.

[3] D. C. Knill and W. Richards. *Perception as Bayesian inference*. Cambridge University Press, 1996.

[4] M. S. Landy, L. T. Maloney, E. B. Johnston, and M. Young. Measurement and modeling of depth cue combination: In defense of weak fusion. *Vision Research*, 35(3):389–412, 1995.

[5] J. Burge, M. A. Peterson, and S. E. Palmer. Ordinal configural cues combine with metric disparity in depth perception. *Journal of Vision*, 5(6):5, 2005.

[6] D. Geiger, B. Ladendorf, and A. Yuille. Occlusions and binocular stereo. *International Journal of Computer Vision*, 14(3):211–226, 1995.

[7] K. Nakayama and S. Shimojo. da vinci stereopsis: Depth and subjective occluding contours from unpaired image points. *Vision Research*, 30(11):1811, 1990.

[8] J. J. Tsai and J. D. Victor. Neither occlusion constraint nor binocular disparity accounts for the perceived depth in the sieve effect. *Vision Research*, 40(17):2265–2275, 2000.

[9] K. P. Körding, U. Beierholm, W. J. Ma, S. Quartz, J. B. Tenenbaum, and L. Shams. Causal inference in multisensory perception. *PLoS ONE*, 2(9), 2007.

[10] K. Wei and K. Körding. Relevance of error: what drives motor adaptation? *Journal of Neurophysiology*, 101(2):655, 2009.

[11] M. O. Ernst and H. H. Bülthoff. Merging the senses into a robust percept. *Trends in Cognitive Sciences*, 8(4):162–169, 2004.

[12] D. C. Knill. Robust cue integration: A bayesian model and evidence from cue-conflict studies with stereoscopic and figure cues to slant. *Journal of Vision*, 7(7):5, 2007.

[13] L. Paninski. Asymptotic theory of information-theoretic experimental design. *Neural Computation*, 17(7):1480–1507, 2005.

[14] K. Nakayama and S. Shimojo. Experiencing and perceiving visual surfaces. *Science*, 257(5075):1357–1363, Sep 1992.

